# A Critical Comparison of Models for Orientation and Ocular Dominance Columns in the Striate Cortex

**E. Erwin**
Beckman Institute
University of Illinois
Urbana, IL 61801, USA

**K. Obermayer**
Technische Fakultät
Universität Bielefeld
33615 Bielefeld, FRG

**K. Schulten**
Beckman Institute
University of Illinois
Urbana, IL 61801, USA

## Abstract

More than ten of the most prominent models for the structure and for the activity dependent formation of orientation and ocular dominance columns in the striate cortex have been evaluated. We implemented those models on parallel machines, we extensively explored parameter space, and we quantitatively compared model predictions with experimental data which were recorded optically from macaque striate cortex.

In our contribution we present a summary of our results to date. Briefly, we find that (*i*) despite apparent differences, many models are based on similar principles and, consequently, make similar predictions, (*ii*) certain "pattern models" as well as the developmental "correlation-based learning" models disagree with the experimental data, and (iii) of the models we have investigated, "competitive Hebbian" models and the recent model of Swindale provide the best match with experimental data.

## 1 Models and Data

The models for the formation and structure of orientation and ocular dominance columns which we have investigated are summarized in table 1. Models fall into two categories: "Pattern models" whose aim is to achieve a concise description of the observed patterns and "developmental models" which are focussed on the pro-

| Class | Type | Model | Reference |
|---|---|---|---|
| Pattern Models | Structural Models | 1. Icecube<br>2. Pinwheel<br>3. Götz<br>4. Baxter | Hubel and Wiesel 1977 [9]<br>Braitenberg and Braitenberg 1979 [6]<br>Götz 1987 [8]<br>Baxter and Dow 1989 [1] |
| | Spectral Models | 5. Rojer<br>6. Niebur<br>7. Swindale | Rojer and Schwartz 1990 [20]<br>Niebur and Wörgötter 1993 [15]<br>Swindale 1992a [21] |
| Develop. Models | Correlation Based Learning | 8. Linsker<br>9. Miller | Linsker 1986c [12]<br>Miller 1989, 1994 [13, 14] |
| | Competitive Hebbian | 10. SOM-h<br>11. SOM-l<br>12. EN | Obermayer, et. al. 1990 [19]<br>Obermayer, et. al. 1992 [17]<br>Durbin and Mitchison 1990 [7] |
| | Other | 13. Tanaka<br>14. Yuille | Tanaka 1991 [22]<br>Yuille, et. al. 1992 [23] |

Table 1: Models of visual cortical maps which have been evaluated.

cesses underlying their formation. Pattern models come in two varieties, "structural models" and "spectral models", which describe orientation and ocular dominance maps in real and in Fourier space, respectively. Developmental models fall into the categories "correlations based learning", "competitive Hebbian" learning and a few miscellaneous models.

Models are compared with data obtained from macaque striate cortex through optical imaging [2, 3, 4, 16]. Data were recorded from the representation of the parafovea from the superficial layers of cortex. In the following we will state that a particular model reproduces a particular feature of the experimental data ($i$) if there exists a parameter regime where the model generates appropriate patterns and ($ii$) if the phenomena are robust. We will state that a particular model does not reproduce a certain feature ($i$) if we have not found an appropriate parameter regime and ($ii$) if there exists either a proof or good intuitive reasons that a model cannot reproduce this feature.

One has to keep in mind, though, that model predictions are compared with a fairly special set of data. Ocular dominance patterns, e.g., are known to vary between species and even between different regions within area 17 of an individual. Consequently, a model which does not reproduce certain features of ocular dominance or orientation columns in the macaque may well describe those patterns in other species. Interspecies differences, however, are not the focus of this contribution; results of corresponding modelling studies will be reported elsewhere.

## 2    Examples of Organizing Principles and Model Predictions

It has been suggested that the most important principles underlying the pattern of orientation and ocular dominance are "continuity" and "diversity" [7, 19, 21]. Continuity, because early image processing is often local in feature space, and diversity, because, e.g., the visual system may want to avoid perceptual scotomata. The continuity and diversity principles underlie almost all descriptive and developmental

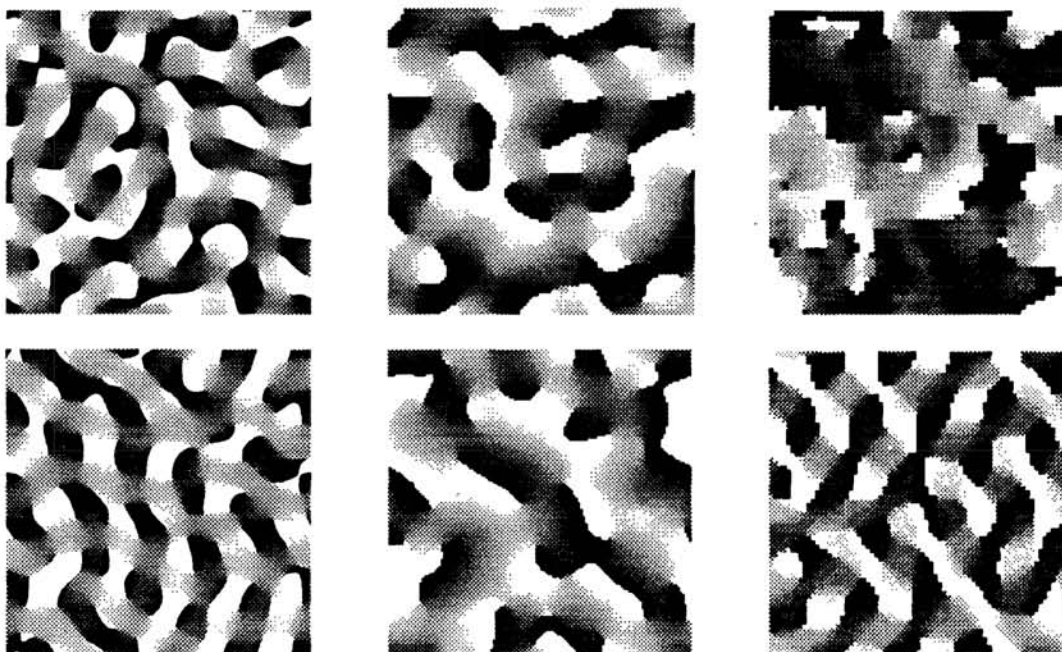

Figure 1: Typical patterns of orientation preferences as they are predicted by six of the models listed in Table 1. Orientation preferences are coded by gray values, where black → white denotes preferences for vertical → horizontal → vertical. **Top row** (left to right): Models 7, 11, 9. **Bottom row** (left to right) Models 5, 12, 8.

models, but maps which comply with these principles often differ in qualitative ways: The icecube model, e.g., obeys both principles but contains no singularities in the orientation preference map and no branching of ocular dominance bands. Figure 1 shows orientation maps generated by six different algorithms taken from Tab. 1. Although all patterns are consistent with the continuity and diversity constraints, closer comparison reveals differences. Thus additional elements of organization must be considered.

It has been suggested that maps are characterized by local correlations and global disorder. Figure 2 (left) shows as an example two-point correlation functions of orientation maps. The autocorrelation function [17] of one of the Cartesian coordinates of the orientation vector is plotted as a function of cortical distance. The fact that all correlation functions decay indicates that the orientation map exhibits global disorder. Global disorder is predicted by all models except the early pattern models 6, 8 and 9. Figure 2 (right) shows the corresponding power spectra. Bandpass-like spectra which are typical for the experimental data [16] are well predicted by models 10–12. Interestingly, they are not predicted by model 9, which also fails reproducing the Mexican-hat shaped correlation functions (bold lines), and model 13.

Based on the fact that experimental maps are characterized by a bandpass-like power spectrum it has been suggested that orientation maps may be organized

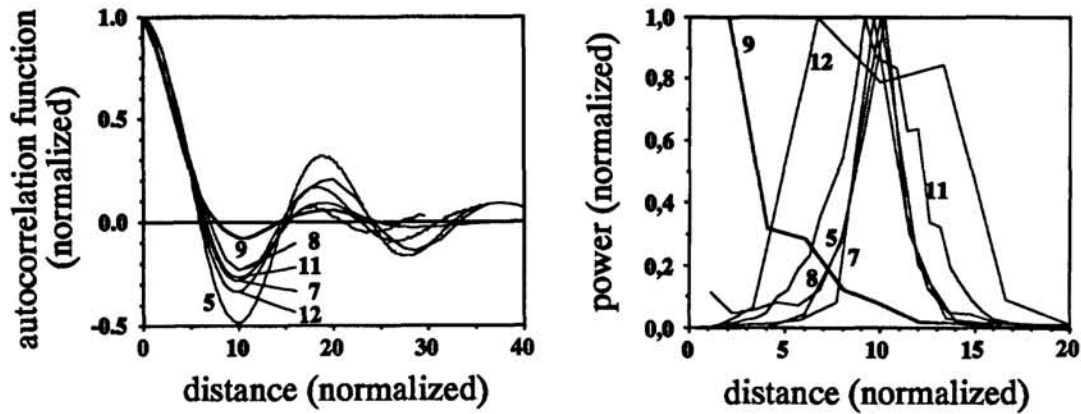

Figure 2: **Left:** Spatial autocorrelation functions for one of the cartesian coordinates of the orientation vector. Autocorrelation functions were averaged over all directions. **Right:** Complex power spectra of orientation maps. Power was averaged over all directions of the wave vector. Model numbers as in Tab. 1.

according to four principles [15]: continuity, diversity, homogeneity and isotropy. If those principles are implemented using bandpass filtered noise the resulting maps [15, 21] indeed share many properties with the experimental data. Above principles alone, however, are not sufficient: (*i*) There are models such as model 5 which are based on those principles but generate different patterns, (*ii*) homogeneity and isotropy are hardly ever fulfilled ([16] and next paragraph), and (*iii*) those principles cannot account for correlations between maps of various response properties [16].

Maps of orientation and ocular dominance in the macaque are anisotropic, i.e., there exist preferred directions along which orientation and ocular dominance slabs align [16]. Those anisotropies can emerge due to different mechanisms: (*i*) spontaneous symmetry breaking, (*ii*) model equations, which are not rotation invariant, and (*iii*) appropriately chosen boundary conditions. Figure 3 illustrates mechanisms (*ii*) and (*iii*) for model 11. Both mechanisms indeed predict anisotropic patterns, however, preferred directions of orientation and ocular dominance align in both cases (fig. 3, left and center). This is not true for the experimental data, where preferred directions tend to be orthogonal [16]. Orthogonal preferred directions can be generated by using different neighborhood functions for different components of the feature vector (fig. 3, right). However, this is not a satisfactory solution, and the issue of anisotropies is still unsolved.

The pattern of orientation preference in the area 17 of the macaque exhibits four local elements of organization: linear zones, singularities, saddle points and fractures [16]. Those elements are correctly predicted by most of the pattern models, except models 1–3, and they appear in the maps generated by models 10–14. Interestingly, models 9 and 13 predict very few linear zones, which is related to the fact that those models generate orientation maps with lowpass-like power spectra.

Another important property of orientation maps is that orientation preferences and their spatial layout across cortex are not correlated which each other. One conse-

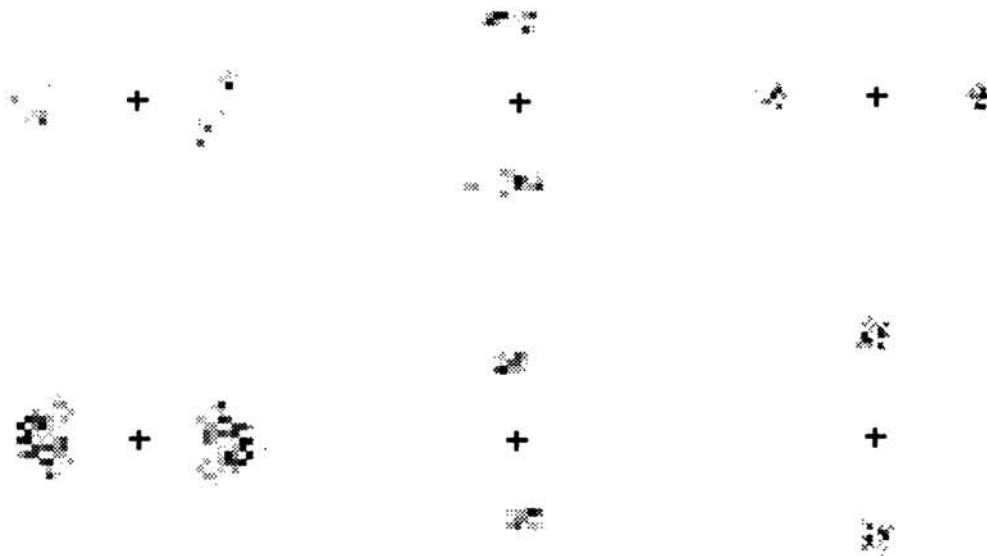

Figure 3: Anisotropic orientation and ocular dominance maps generated by model 11. The figure shows Fourier spectra [17] of orientation (**top row**) and ocular dominance maps (**bottom row**). **Left:** Maps generated with an elliptic neighborhood function (case (*ii*), see text); **Center:** Maps generated using circular input layers and an elliptical cortical sheet (case (*iii*), see text), **Right:** Maps generated with different, elliptic neighborhood functions for orientation preference and ocular dominance. '+' symbols indicate the locations of the origin.

quence is that there exist singularities, near which the curl of the orientation vector field does not vanish (fig. 4, left). This rules out a class of pattern models where the orientation map is derived from the gradient of a potential function, model 5. Figure 4 (right) shows another consequence of this property. In those figures cortical area is plotted against the angular difference between the iso-orientation lines and the local orientation preference. The even distribution found in the experimental data is correctly predicted by models 1, 6, 7 and 10–12. Model 8, however, predicts preference for large difference angles while model 9 - over a wide range of parameters - predicts preference for small difference angles (bold lines).

Finally, let us consider correlations between the patterns of orientation preference and ocular dominance. Among the more prominent relationships present in macaque data are [3, 16, 21]: (*i*) Singularities are aligned with the centers of ocular dominance bands, (*ii*) fractures are either aligned or run perpendicular, and (*iii*) iso-orientation bands in linear zones intersect ocular dominance bands at approximately right angles. Those relationships are readily reproduced only by models 7 and 10–12. For model 9 reasonable orientation and ocular dominance patterns have not been generated at the same time. It would seem as if the parameter regime where reasonable orientation columns emerge is incompatible with the parameter regime where ocular dominance patterns are formed.

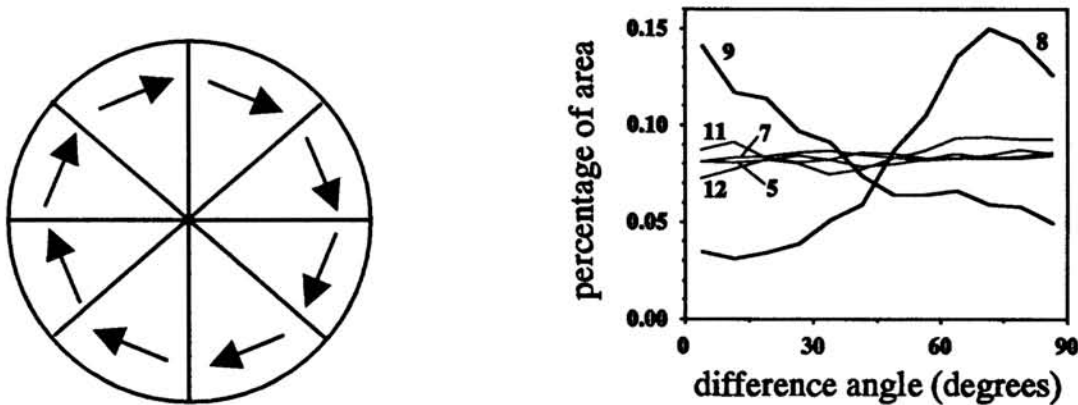

Figure 4: **Left**: This singularity is an example of a feature in the experimental data which is not allowed by model 5. The arrows indicate orientation vectors, whose angular component is twice the value of the local orientation preference. **Right**: Percentage of area as a function of the angular difference between preferred orientation and the local orientation gradient vector. Model numbers as in Table 1.

## 3    The Current Status of the Model Comparison Project

Lack of space prohibits a detailed discussion of our findings but we have summarized the current status of our project in Tables 2 and 3. Given the models listed in Tab. 1 and given the properties of the orientation and ocular dominance patterns in macaque striate cortex listed in Tables 2 and 3 it is models 7 and 10–12 which currently are in best agreement with the data. Those models, however, are fairly abstract and simplified, and they cannot easily be extended to predict receptive field structure. Biological realism and predictions about receptive fields are the advantages of models 8 and 9. Those models, however, cannot account for the observed orientation patterns. It would, therefore, be of high interest, if elements of both approaches could be combined to achieve a better description of the data.

The main conclusion, however, is that there are now enough data available to allow a better evaluation of model approaches than just by visual comparison of the generated patterns. It is our hope, that future studies will address at least those properties of the patterns which are known and well described, some of which are listed in Tables 2 and 3. In case of developmental models more stringent tests require experiments which (*i*) monitor the actual time-course of pattern formation, and which (*ii*) study pattern development under experimentally modified conditions (deprivation experiments). Currently there is not enough data available to constrain models but the experiments are under way [5, 10, 11, 18].

### Acknowledgements

We are very much indebted to Drs. Linsker, Tanaka and Yuille for sharing modelling data. E.E. thanks the Beckman Institute for support. K.O. thanks ZiF (Universität Bielefeld) for its hospitality. Computing time on a CM-2 and a CM-5 was made available by NCSA.

| | | | | | Properties of OR Maps | | | | | |
|---|---|---|---|---|---|---|---|---|---|---|
| no. | dis-order | band-pass | linear zones | saddle points | sing. $\pm 1/2$ | fract. | indep. coord. | high spec. | aniso-tropy | OR-bias |
| 1 | - | + | + | - | - | - | + | n | + | n |
| 2 | - | + | + | + | - | - | - | n | n | n |
| 3 | - | + | + | + | + | - | - | n | n | n |
| 4 | $+^2$ | + | + | + | $+^2$ | - | - | n | + | n |
| 5 | + | + | + | + | + | $+^1$ | - | - | + | n |
| 6 | + | + | + | + | + | $+^1$ | + | - | + | n |
| 7 | + | + | + | + | + | $+^1$ | + | + | + | n |
| 8 | + | + | - | + | + | + | - | n | n | n |
| 9 | + | - | - | + | + | + | -/+ | + | n | n |
| 10 | + | + | - | + | + | $+^1$ | + | + | + | + |
| 11 | + | + | + | + | + | $+^1$ | + | + | + | + |
| 12 | + | + | + | + | + | $+^1$ | + | + | + | + |
| 13 | + | - | + | + | + | $+^1$ | + | + | n | n |
| 14 | + | ? | ? | + | + | + | ? | n | n | n |

Table 2: Evaluation of orientation (OR) map models. Properties of the experimental maps include (left to right): global disorder; bandpass-like power spectra; the presence of linear zones in roughly 50% of the map area; the presence of saddle points, singularities ($\pm 1/2$ with equal densities), and fractures; independence between cortical and orientation preference coordinates; a distribution favoring high values of orientation specificity; global anisotropy; and a possible orientation bias. Symbols: '+': There exists a parameter regime in which a model generates maps with this property; '-': The model cannot reproduce this property; 'n': The model makes no predictions; '?': Not enough data available. [1]Models agree with the data only if one assumes that fractures are loci of rapid orientation change rather than real discontinuities. [2]One of several cases.

# References

[1] W. T. Baxter and B. M. Dow. *Biol. Cybern.*, 61:171–182, 1989.

[2] G. G. Blasdel. *J. Neurosci.*, 12:3115–3138, 1992.

[3] G. G. Blasdel. *J. Neurosci.*, 12:3139–3161,1992.

[4] G. G. Blasdel and G. Salama. *Nature*, 321:579–585, 1986.

[5] T. Bonhoeffer, D. Kim, and W. Singer. *Soc. Neurosci. Abs.*, 19:1800, 1993.

[6] V. Braitenberg and C. Braitenberg. *Biol. Cybern.*, 33:179–186, 1979.

[7] R. Durbin and G. Mitchison. *Nature*, 343:341–344, 1990.

[8] K. G. Götz. *Biol. Cybern.*, 56:107–109, 1987.

[9] D. Hubel and T. N. Wiesel. *Proc. Roy. Soc. Lond. B*, 198:1–59, 1977.

[10] D. Hubel, T. N. Wiesel, and S. LeVay. *Phil. Trans. Roy. Soc. Lond. B*, 278:377–409, 1977.

[11] D. Kim and T. Bonhoeffer. *Soc. Neurosci. Abs.*, 19:1800, 1993.

[12] R. Linsker. *Proc. Nat. Acad. Sci., USA*, 83:8779–8783, 1986.

| no. | Properties of OD Maps | | | | | Correlations Between OR and OD | | | |
|---|---|---|---|---|---|---|---|---|---|
|  | segregation | disorder | anisotropy | OD-bias | strabismus | local orthog. | global orthog. | sing. vs. OD | spec. vs. OD |
| 1 | + | - | + | + | n | +[2] | +[2] | - | n |
| 2 | n | n | n | n | n | n | n | n | n |
| 3 | + | - | + | n | n | + | n | +[2] | n |
| 4 | n | n | n | n | n | n | n | n | n |
| 5 | + | + | + | - | n | +[1] | -[1] | +[1,2] | -[1] |
| 6 | n | n | n | n | n | n | n | n | n |
| 7 | + | + | + | + | n | - | + | + | + |
| 8 | n | n | n | n | n | n | n | n | n |
| 9 | + | + | + | + | + | ?[1] | ?[1] | ?[1] | ?[1] |
| 10 | + | + | + | + | + | + | n | + | + |
| 11 | + | + | + | + | + | +[2] | n | +[2] | +[2] |
| 12 | + | + | + | + | + | +[1,2] | n | +[1,2] | +[1,2] |
| 13 | + | + | + | + | + | n | n | n | n |
| 14 | + | + | + | + | n | n | n | n | n |

Table 3: **Left**: Evaluation of ocular dominance (OD) map models. Properties of the experimental maps include (left to right): Segregated bands of eye dominance; global disorder; bandpass-like power spectra; global anisotropy; a bias to the representation of one eye; and OD-patterns in animals with strabismus. **Right**: Evaluation of correlations between OD and OR. Experimental maps show (left to right): Local and global orthogonality between OR and OD slabs; singularities preferably in monocular regions, and lower OR specificity in monocular regions. [1]Authors treated OD and OR in independent models, but we consider a combined version. [2]Correlations are stronger than in the experimental data.

[13] K. D. Miller. *J. Neurosci.*, 14:409–441, 1994.

[14] K. D. Miller, J. B. Keller, and M. P. Stryker. *Science*, 245:605–615, 1989.

[15] E. Niebur and F. Wörgötter. In F. H. Eeckman and J. M. Bower, *Computation and Neural Systems*, pp. 409–413. Kluwer Academic Publishers, 1993.

[16] K. Obermayer and G. G. Blasdel. *J. Neurosci.*, 13:4114–4129, 1993.

[17] K. Obermayer, G. G. Blasdel, and K. Schulten. *Phys. Rev. A*, 45:7568–7589, 1992.

[18] K. Obermayer, L. Kiorpes, and G. G. Blasdel. In J. D. Cowan at al., *Advances in Neural Information Processing Systems 6*. Morgan Kaufmann, 1994. 543-550.

[19] K. Obermayer, H. Ritter, and K. Schulten. *Proc. Nat. Acad. Sci., USA*, 87:8345–8349, 1990.

[20] A. S. Rojer and E. L. Schwartz. *Biol. Cybern.*, 62:381–391, 1990.

[21] N. V. Swindale. *Biol. Cybern.*, 66:217–230, 1992.

[22] S. Tanaka. *Biol. Cybern.*, 65:91–98, 1991.

[23] A. L. Yuille, J. A. Kolodny, and C. W. Lee. TR 91-3, Harvard Robotics Laboratory, 1991.
